# Map-Reduce for Machine Learning on Multicore

**Cheng-Tao Chu** *
chengtao@stanford.edu

**Sang Kyun Kim** *
skkim38@stanford.edu

**Yi-An Lin** *
ianl@stanford.edu

**YuanYuan Yu** *
yuanyuan@stanford.edu

**Gary Bradski** *†
garybradski@gmail

**Andrew Y. Ng** *
ang@cs.stanford.edu

**Kunle Olukotun** *
kunle@cs.stanford.edu

*. CS. Department, Stanford University 353 Serra Mall,
Stanford University, Stanford CA 94305-9025.
† Rexee Inc.

## Abstract

We are at the beginning of the multicore era. Computers will have increasingly many cores (processors), but there is still no good programming framework for these architectures, and thus no simple and unified way for machine learning to take advantage of the potential speed up. In this paper, we develop a broadly applicable parallel programming method, one that is easily applied to *many* different learning algorithms. Our work is in distinct contrast to the tradition in machine learning of designing (often ingenious) ways to speed up a *single* algorithm at a time. Specifically, we show that algorithms that fit the Statistical Query model [15] can be written in a certain "summation form," which allows them to be easily parallelized on multicore computers. We adapt Google's map-reduce [7] paradigm to demonstrate this parallel speed up technique on a variety of learning algorithms including locally weighted linear regression (LWLR), k-means, logistic regression (LR), naive Bayes (NB), SVM, ICA, PCA, gaussian discriminant analysis (GDA), EM, and backpropagation (NN). Our experimental results show basically linear speedup with an increasing number of processors.

## 1 Introduction

Frequency scaling on silicon—the ability to drive chips at ever higher clock rates—is beginning to hit a power limit as device geometries shrink due to leakage, and simply because CMOS consumes power every time it changes state [9, 10]. Yet Moore's law [20], the density of circuits doubling every generation, is projected to last between 10 and 20 more years for silicon based circuits [10]. By keeping clock frequency fixed, but doubling the number of processing cores on a chip, one can maintain lower power while doubling the speed of many applications. This has forced an industry-wide shift to multicore.

We thus approach an era of increasing numbers of cores per chip, but there is as yet no good framework for machine learning to take advantage of massive numbers of cores. There are many parallel programming languages such as Orca, Occam ABCL, SNOW, MPI and PARLOG, but none of these approaches make it obvious how to parallelize a particular algorithm. There is a vast literature on distributed learning and data mining [18], but very little of this literature focuses on our goal: A general means of programming machine learning on multicore. Much of this literature contains a long

and distinguished tradition of developing (often ingenious) ways to speed up or parallelize *individual* learning algorithms, for instance cascaded SVMs [11]. But these yield no general parallelization technique for machine learning and, more pragmatically, specialized implementations of popular algorithms rarely lead to widespread use. Some examples of more general papers are: Caregea et. al. [5] give some general data distribution conditions for parallelizing machine learning, but restrict the focus to decision trees; Jin and Agrawal [14] give a general machine learning programming approach, but only for shared memory machines. This doesn't fit the architecture of cellular or grid type multiprocessors where cores have local cache, even if it can be dynamically reallocated.

In this paper, we focuses on developing a general and exact technique for parallel programming of a large class of machine learning algorithms for multicore processors. The central idea of this approach is to allow a future programmer or user to speed up machine learning applications by "throwing more cores" at the problem rather than search for specialized optimizations. This paper's contributions are:

$(i)$ We show that any algorithm fitting the Statistical Query Model may be written in a certain "summation form." This form does not change the underlying algorithm and so is not an approximation, but is instead an exact implementation. $(ii)$ The summation form does not depend on, but can be easily expressed in a map-reduce [7] framework which is easy to program in. $(iii)$ This technique achieves basically linear speed-up with the number of cores.

We attempt to develop a pragmatic and general framework. What we do **not** claim:

$(i)$ We make no claim that our technique will necessarily run faster than a specialized, one-off solution. Here we achieve linear speedup which in fact often does beat specific solutions such as cascaded SVM [11] (see section 5; however, they do handle kernels, which we have not addressed). $(ii)$ We make no claim that following our framework (for a *specific* algorithm) always leads to a novel parallelization undiscovered by others. What is novel is the larger, broadly applicable framework, together with a pragmatic programming paradigm, map-reduce. $(iii)$ We focus here on exact implementation of machine learning algorithms, not on parallel approximations to algorithms (a worthy topic, but one which is beyond this paper's scope).

In section 2 we discuss the Statistical Query Model, our summation form framework and an example of its application. In section 3 we describe how our framework may be implemented in a Google-like map-reduce paradigm. In section 4 we choose 10 frequently used machine learning algorithms as examples of what can be coded in this framework. This is followed by experimental runs on 10 moderately large data sets in section 5, where we show a good match to our theoretical computational complexity results. Basically, we often achieve linear speedup in the number of cores. Section 6 concludes the paper.

## 2   Statistical Query and Summation Form

For multicore systems, Sutter and Larus [25] point out that multicore mostly benefits concurrent applications, meaning ones where there is little communication between cores. The best match is thus if the data is subdivided and stays local to the cores. To achieve this, we look to Kearns' Statistical Query Model [15].

The Statistical Query Model is sometimes posed as a restriction on the Valiant PAC model [26], in which we permit the learning algorithm to access the learning problem only through a *statistical query oracle*. Given a function $f(x, y)$ over instances, the statistical query oracle returns an estimate of the expectation of $f(x, y)$ (averaged over the training/test distribution). Algorithms that calculate sufficient statistics or gradients fit this model, and since these calculations may be batched, they are expressible as a sum over data points. This class of algorithms is large; We show 10 popular algorithms in section 4 below. An example that does not fit is that of learning an XOR over a subset of bits. [16, 15]. However, when an algorithm does sum over the data, we can easily distribute the calculations over multiple cores: We just divide the data set into as many pieces as there are cores, give each core its share of the data to sum the equations over, and aggregate the results at the end. We call this form of the algorithm the "summation form."

As an example, consider ordinary least squares (linear regression), which fits a model of the form $y = \theta^T x$ by solving: $\theta^* = \min_\theta \sum_{i=1}^m (\theta^T x_i - y_i)^2$ The parameter $\theta$ is typically solved for by

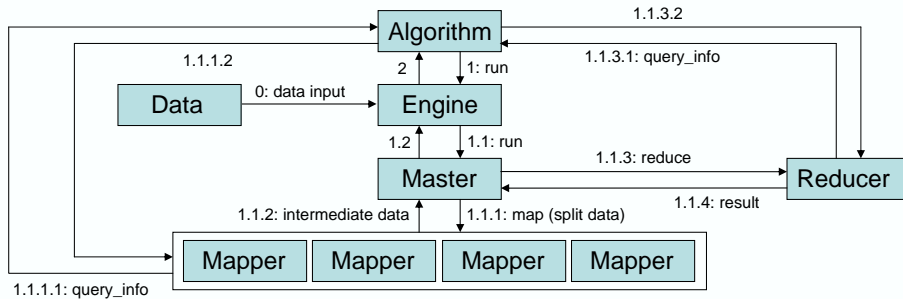

Figure 1: Multicore map-reduce framework

defining the design matrix $X \in \mathbb{R}^{m \times n}$ to be a matrix whose rows contain the training instances $x_1, \ldots, x_m$, letting $\vec{y} = [y_1, \ldots, y_m]^m$ be the vector of target labels, and solving the normal equations to obtain $\theta^* = (X^T X)^{-1} X^T \vec{y}$.

To put this computation into summation form, we reformulate it into a two phase algorithm where we first compute sufficient statistics by summing over the data, and then aggregate those statistics and solve to get $\theta^* = A^{-1} b$. Concretely, we compute $A = X^T X$ and $b = X^T \vec{y}$ as follows: $A = \sum_{i=1}^{m} (x_i x_i^T)$ and $b = \sum_{i=1}^{m} (x_i y_i)$. The computation of $A$ and $b$ can now be divided into equal size pieces and distributed among the cores. We next discuss an architecture that lends itself to the summation form: Map-reduce.

## 3 Architecture

Many programming frameworks are possible for the summation form, but inspired by Google's success in adapting a functional programming construct, map-reduce [7], for wide spread parallel programming use inside their company, we adapted this same construct for multicore use. Google's map-reduce is specialized for use over clusters that have unreliable communication and where individual computers may go down. These are issues that multicores do not have; thus, we were able to developed a much lighter weight architecture for multicores, shown in Figure 1.

Figure 1 shows a high level view of our architecture and how it processes the data. In step 0, the map-reduce engine is responsible for splitting the data by training examples (rows). The engine then caches the split data for the subsequent map-reduce invocations. Every algorithm has its own engine instance, and every map-reduce task will be delegated to its engine (step 1). Similar to the original map-reduce architecture, the engine will run a master (step 1.1) which coordinates the mappers and the reducers. The master is responsible for assigning the split data to different mappers, and then collects the processed intermediate data from the mappers (step 1.1.1 and 1.1.2). After the intermediate data is collected, the master will in turn invoke the reducer to process it (step 1.1.3) and return final results (step 1.1.4). Note that some mapper and reducer operations require additional scalar information from the algorithms. In order to support these operations, the mapper/reducer can obtain this information through the query_info interface, which can be customized for each different algorithm (step 1.1.1.1 and 1.1.3.2).

## 4 Adopted Algorithms

In this section, we will briefly discuss the algorithms we have implemented based on our framework. These algorithms were chosen partly by their popularity of use in NIPS papers, and our goal will be to illustrate how each algorithm can be expressed in summation form. We will defer the discussion of the theoretical improvement that can be achieved by this parallelization to Section 4.1. In the following, $x$ or $x_i$ denotes a training vector and $y$ or $y_i$ denotes a training label.

- **Locally Weighted Linear Regression (LWLR)** LWLR [28, 3] is solved by finding the solution of the normal equations $A\theta = b$, where $A = \sum_{i=1}^{m} w_i(x_i x_i^T)$ and $b = \sum_{i=1}^{m} w_i(x_i y_i)$. For the summation form, we divide the computation among different mappers. In this case, one set of mappers is used to compute $\sum_{subgroup} w_i(x_i x_i^T)$ and another set to compute $\sum_{subgroup} w_i(x_i y_i)$. Two reducers respectively sum up the partial values for $A$ and $b$, and the algorithm finally computes the solution $\theta = A^{-1}b$. Note that if $w_i = 1$, the algorithm reduces to the case of ordinary least squares (linear regression).

- **Naive Bayes (NB)** In NB [17, 21], we have to estimate $P(x_j = k|y = 1)$, $P(x_j = k|y = 0)$, and $P(y)$ from the training data. In order to do so, we need to sum over $x_j = k$ for each $y$ label in the training data to calculate $P(x|y)$. We specify different sets of mappers to calculate the following: $\sum_{subgroup} 1\{x_j = k|y = 1\}$, $\sum_{subgroup} 1\{x_j = k|y = 0\}$, $\sum_{subgroup} 1\{y = 1\}$ and $\sum_{subgroup} 1\{y = 0\}$. The reducer then sums up intermediate results to get the final result for the parameters.

- **Gaussian Discriminative Analysis (GDA)** The classic GDA algorithm [13] needs to learn the following four statistics $P(y), \mu_0, \mu_1$ and $\Sigma$. For all the summation forms involved in these computations, we may leverage the map-reduce framework to parallelize the process. Each mapper will handle the summation (i.e. $\Sigma\ 1\{y_i = 1\}, \Sigma\ 1\{y_i = 0\}, \Sigma\ 1\{y_i = 0\}x_i$, etc) for a subgroup of the training samples. Finally, the reducer will aggregate the intermediate sums and calculate the final result for the parameters.

- **k-means** In k-means [12], it is clear that the operation of computing the Euclidean distance between the sample vectors and the centroids can be parallelized by splitting the data into individual subgroups and clustering samples in each subgroup separately (by the mapper). In recalculating new centroid vectors, we divide the sample vectors into subgroups, compute the sum of vectors in each subgroup in parallel, and finally the reducer will add up the partial sums and compute the new centroids.

- **Logistic Regression (LR)** For logistic regression [23], we choose the form of hypothesis as $h_\theta(x) = g(\theta^T x) = 1/(1 + \exp(-\theta^T x))$ Learning is done by fitting $\theta$ to the training data where the likelihood function can be optimized by using Newton-Raphson to update $\theta := \theta - H^{-1}\nabla_\theta \ell(\theta)$. $\nabla_\theta \ell(\theta)$ is the gradient, which can be computed in parallel by mappers summing up $\sum_{subgroup}(y^{(i)} - h_\theta(x^{(i)}))x_j^{(i)}$ each NR step $i$. The computation of the hessian matrix can be also written in a summation form of $H(j,k) := H(j,k) + h_\theta(x^{(i)})(h_\theta(x^{(i)}) - 1)x_j^{(i)}x_k^{(i)}$ for the mappers. The reducer will then sum up the values for gradient and hessian to perform the update for $\theta$.

- **Neural Network (NN)** We focus on backpropagation [6] By defining a network structure (we use a three layer network with two output neurons classifying the data into two categories), each mapper propagates its set of data through the network. For each training example, the error is back propagated to calculate the partial gradient for each of the weights in the network. The reducer then sums the partial gradient from each mapper and does a batch gradient descent to update the weights of the network.

- **Principal Components Analysis (PCA)** PCA [29] computes the principle eigenvectors of the covariance matrix $\Sigma = \frac{1}{m}\left(\sum_{i=1}^{m} x_i x_i^T\right) - \mu\mu^T$ over the data. In the definition for $\Sigma$, the term $\left(\sum_{i=1}^{m} x_i x_i^T\right)$ is already expressed in summation form. Further, we can also express the mean vector $\mu$ as a sum, $\mu = \frac{1}{m}\sum_{i=1}^{m} x_i$. The sums can be mapped to separate cores, and then the reducer will sum up the partial results to produce the final empirical covariance matrix.

- **Independent Component Analysis (ICA)** ICA [1] tries to identify the independent source vectors based on the assumption that the observed data are linearly transformed from the source data. In ICA, the main goal is to compute the unmixing matrix W. We implement batch gradient ascent to optimize the $W$'s likelihood. In this scheme, we can independently calculate the expression $\begin{bmatrix} 1 - 2g(w_1^T x^{(i)}) \\ \vdots \end{bmatrix} x^{(i)T}$ in the mappers and sum them up in the reducer.

- **Expectation Maximization (EM)** For EM [8] we use Mixture of Gaussian as the underlying model as per [19]. For parallelization: In the E-step, every mapper processes its subset

of the training data and computes the corresponding $w_j^{(i)}$ (expected pseudo count). In M-phase, three sets of parameters need to be updated: $p(y)$, $\mu$, and $\Sigma$. For $p(y)$, every mapper will compute $\sum_{subgroup}(w_j^{(i)})$, and the reducer will sum up the partial result and divide it by $m$. For $\mu$, each mapper will compute $\sum_{subgroup}(w_j^{(i)} * x^{(i)})$ and $\sum_{subgroup}(w_j^{(i)})$, and the reducer will sum up the partial result and divide them. For $\Sigma$, every mapper will compute $\sum_{subgroup}(w_j^{(i)} * (x^{(i)} - \mu_j) * (x^{(i)} - \mu_j)^T)$ and $\sum_{subgroup}(w_j^{(i)})$, and the reducer will again sum up the partial result and divide them.

- **Support Vector Machine (SVM)** Linear SVM's [27, 22] primary goal is to optimize the following primal problem $\min_{w,b} \|w\|^2 + C \sum_{i:\xi_i>0} \xi_i^p \quad s.t. \quad y^{(i)}(w^T x^{(i)} + b) \geq 1 - \xi_i$ where $p$ is either 1 (hinge loss) or 2 (quadratic loss). [2] has shown that the primal problem for quadratic loss can be solved using the following formula where $sv$ are the support vectors: $\nabla = 2w + 2C \sum_{i \in sv} (w \cdot x_i - y_i) x_i$ & Hessian $H = I + C \sum_{i \in sv} x_i x_i^T$ We perform batch gradient descent to optimize the objective function. The mappers will calculate the partial gradient $\sum_{subgroup(i \in sv)} (w \cdot x_i - y_i) x_i$ and the reducer will sum up the partial results to update $w$ vector.

Some implementations of machine learning algorithms, such as ICA, are commonly done with stochastic gradient ascent, which poses a challenge to parallelization. The problem is that in every step of gradient ascent, the algorithm updates a common set of parameters (e.g. the unmixing $W$ matrix in ICA). When one gradient ascent step (involving one training sample) is updating $W$, it has to lock down this matrix, read it, compute the gradient, update $W$, and finally release the lock. This "lock-release" block creates a bottleneck for parallelization; thus, instead of stochastic gradient ascent, our algorithms above were implemented using batch gradient ascent.

## 4.1 Algorithm Time Complexity Analysis

Table 1 shows the theoretical complexity analysis for the ten algorithms we implemented on top of our framework. We assume that the dimension of the inputs is $n$ (i.e., $x \in \mathbb{R}^n$), that we have $m$ training examples, and that there are $P$ cores. The complexity of iterative algorithms is analyzed for one iteration, and so their actual running time may be slower.[1] A few algorithms require matrix inversion or an eigen-decomposition of an $n$-by-$n$ matrix; we did not parallelize these steps in our experiments, because for us $m >> n$, and so their cost is small. However, there is extensive research in numerical linear algebra on parallelizing these numerical operations [4], and in the complexity analysis shown in the table, we have assumed that matrix inversion and eigen-decompositions can be sped up by a factor of $P'$ on $P$ cores. (In practice, we expect $P' \approx P$.) In our own software implementation, we had $P' = 1$. Further, the reduce phase can minimize communication by combining data as it's passed back; this accounts for the $\log(P)$ factor.

As an example of our running-time analysis, for single-core LWLR we have to compute $A = \sum_{i=1}^m w_i(x_i x_i^T)$, which gives us the $mn^2$ term. This matrix must be inverted for $n^3$; also, the reduce step incurs a covariance matrix communication cost of $n^2$.

## 5 Experiments

To provide fair comparisons, each algorithm had two different versions: One running map-reduce, and the other a serial implementation without the framework. We conducted an extensive series of experiments to compare the speed up on data sets of various sizes (table 2), on eight commonly used machine learning data sets from the UCI Machine Learning repository and two other ones from a [anonymous] research group (Helicopter Control and sensor data). Note that not all the experiments make sense from an output view – regression on categorical data – but our purpose was to test speedup so we ran every algorithm over all the data.

The first environment we conducted experiments on was an Intel X86 PC with two Pentium-III 700 MHz CPUs and 1GB physical memory. The operating system was Linux RedHat 8.0 Kernel 2.4.20-

|        | single | multi |
|--------|--------|-------|
| LWLR   | $O(mn^2 + n^3)$ | $O(\frac{mn^2}{P} + \frac{n^3}{P'} + n^2 \log(P))$ |
| LR     | $O(mn^2 + n^3)$ | $O(\frac{mn^2}{P} + \frac{n^3}{P'} + n^2 \log(P))$ |
| NB     | $O(mn + nc)$ | $O(\frac{mn}{P} + nc \log(P))$ |
| NN     | $O(mn + nc)$ | $O(\frac{mn}{P} + nc \log(P))$ |
| GDA    | $O(mn^2 + n^3)$ | $O(\frac{mn^2}{P} + \frac{n^3}{P'} + n^2 \log(P))$ |
| PCA    | $O(mn^2 + n^3)$ | $O(\frac{mn^2}{P} + \frac{n^3}{P'} + n^2 \log(P))$ |
| ICA    | $O(mn^2 + n^3)$ | $O(\frac{mn^2}{P} + \frac{n^3}{P'} + n^2 \log(P))$ |
| k-means | $O(mnc)$ | $O(\frac{mnc}{P} + mn \log(P))$ |
| EM     | $O(mn^2 + n^3)$ | $O(\frac{mn^2}{P} + \frac{n^3}{P'} + n^2 \log(P))$ |
| SVM    | $O(m^2 n)$ | $O(\frac{m^2 n}{P} + n \log(P))$ |

Table 1: time complexity analysis

| Data Sets | samples (m) | features (n) |
|-----------|-------------|--------------|
| Adult | 30162 | 14 |
| Helicopter Control | 44170 | 21 |
| Corel Image Features | 68040 | 32 |
| IPUMS Census | 88443 | 61 |
| Synthetic Time Series | 100001 | 10 |
| Census Income | 199523 | 40 |
| ACIP Sensor | 229564 | 8 |
| KDD Cup 99 | 494021 | 41 |
| Forest Cover Type | 581012 | 55 |
| 1990 US Census | 2458285 | 68 |

Table 2: data sets size and description

8smp. In addition, we also ran extensive comparison experiments on a 16 way Sun Enterprise 6000, running Solaris 10; here, we compared results using 1,2,4,8, and 16 cores.

## 5.1 Results and Discussion

Table 3 shows the speedup on dual processors over all the algorithms on all the data sets. As can be seen from the table, most of the algorithms achieve more than 1.9x times performance improvement. For some of the experiments, e.g. gda/covertype, ica/ipums, nn/colorhistogram, etc., we obtain a greater than 2x speedup. This is because the original algorithms do not utilize all the cpu cycles efficiently, but do better when we distribute the tasks to separate threads/processes.

Figure 2 shows the speedup of the algorithms over all the data sets for 2,4,8 and 16 processing cores. In the figure, the thick lines shows the average speedup, the error bars show the maximum and minimum speedups and the dashed lines show the variance. Speedup is basically linear with number

|               | lwlr  | gda   | nb    | logistic | pca   | ica   | svm   | nn    | kmeans | em    |
|---------------|-------|-------|-------|----------|-------|-------|-------|-------|--------|-------|
| Adult         | 1.922 | 1.801 | 1.844 | 1.962    | 1.809 | 1.857 | 1.643 | 1.825 | 1.947  | 1.854 |
| Helicopter    | 1.93  | 2.155 | 1.924 | 1.92     | 1.791 | 1.856 | 1.744 | 1.847 | 1.857  | 1.86  |
| Corel Image   | 1.96  | 1.876 | 2.002 | 1.929    | 1.97  | 1.936 | 1.754 | 2.018 | 1.921  | 1.832 |
| IPUMS         | 1.963 | 2.23  | 1.965 | 1.938    | 1.965 | 2.025 | 1.799 | 1.974 | 1.957  | 1.984 |
| Synthetic     | 1.909 | 1.964 | 1.972 | 1.92     | 1.842 | 1.907 | 1.76  | 1.902 | 1.888  | 1.804 |
| Census Income | 1.975 | 2.179 | 1.967 | 1.941    | 2.019 | 1.941 | 1.88  | 1.896 | 1.961  | 1.99  |
| Sensor        | 1.927 | 1.853 | 2.01  | 1.913    | 1.955 | 1.893 | 1.803 | 1.914 | 1.953  | 1.949 |
| KDD           | 1.969 | 2.216 | 1.848 | 1.927    | 2.012 | 1.998 | 1.946 | 1.899 | 1.973  | 1.979 |
| Cover Type    | 1.961 | 2.232 | 1.951 | 1.935    | 2.007 | 2.029 | 1.906 | 1.887 | 1.963  | 1.991 |
| Census        | 2.327 | 2.292 | 2.008 | 1.906    | 1.997 | 2.001 | 1.959 | 1.883 | 1.946  | 1.977 |
| avg.          | 1.985 | 2.080 | 1.950 | 1.930    | 1.937 | 1.944 | 1.819 | 1.905 | 1.937  | 1.922 |

Table 3: Speedups achieved on a dual core processor, without load time. Numbers reported are dual-core time / single-core time. Super linear speedup sometimes occurs due to a reduction in processor idle time with multiple threads.

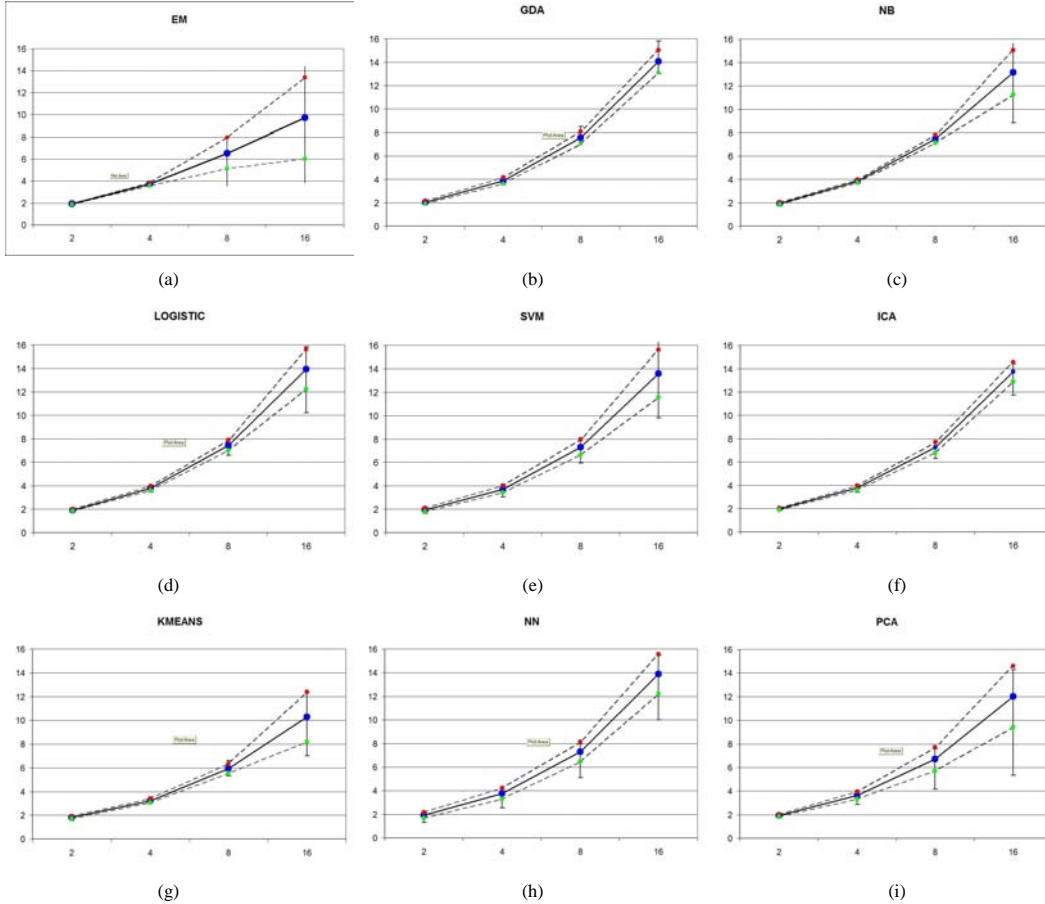

Figure 2: (a)-(i) show the speedup from 1 to 16 processors of all the algorithms over all the data sets. The Bold line is the average, error bars are the max and min speedups and the dashed lines are the variance.

of cores, but with a slope $< 1.0$. The reason for the sub-unity slope is increasing communication overhead. For simplicity and because the number of data points $m$ typically dominates reduction phase communication costs (typically a factor of $n^2$ but $n << m$), we did not parallelize the reduce phase where we could have combined data on the way back. Even so, our simple SVM approach gets about 13.6% speed up on average over 16 cores whereas the specialized SVM cascade [11] averages only 4%.

Finally, the above are runs on multiprocessor machines. We finish by reporting some confirming results and higher performance on a proprietary multicore simulator over the sensor dataset.[2] NN speedup was [16 cores, 15.5x], [32 cores, 29x], [64 cores, 54x]. LR speedup was [16 cores, 15x], [32 cores, 29.5x], [64 cores, 53x]. Multicore machines are generally faster than multiprocessor machines because communication internal to the chip is much less costly.

## 6   Conclusion

As the Intel and AMD product roadmaps indicate [24], the number of processing cores on a chip will be doubling several times over the next decade, even as individual cores cease to become significantly faster. For machine learning to continue reaping the bounty of Moore's law and apply to ever larger datasets and problems, it is important to adopt a programming architecture which takes advantage of multicore. In this paper, by taking advantage of the summation form in a map-reduce

framework, we could parallelize a wide range of machine learning algorithms and achieve a 1.9 times speedup on a dual processor on up to 54 times speedup on 64 cores. These results are in line with the complexity analysis in Table 1. We note that the speedups achieved here involved no special optimizations of the algorithms themselves. We have demonstrated a simple programming framework where in the future we can just "throw cores" at the problem of speeding up machine learning code.

## Acknowledgments

We would like to thank Skip Macy from Intel for sharing his valuable experience in VTune performance analyzer. Yirong Shen, Anya Petrovskaya, and Su-In Lee from Stanford University helped us in preparing various data sets used in our experiments. This research was sponsored in part by the Defense Advanced Research Projects Agency (DARPA) under the ACIP program and grant number NBCH104009.

## Footnotes

[1]If, for example, the number of iterations required grows with $m$. However, this would affect single- and multi-core implementations equally.

[2]This work was done in collaboration with Intel Corporation.

## References

[1] Sejnowski TJ. Bell AJ. An information-maximization approach to blind separation and blind deconvolution. In *Neural Computation*, 1995.

[2] O. Chapelle. Training a support vector machine in the primal. *Journal of Machine Learning Research (submitted)*, 2006.

[3] W. S. Cleveland and S. J. Devlin. Locally weighted regression: An approach to regression analysis by local fitting. In *J. Amer. Statist. Assoc. 83*, pages 596–610, 1988.

[4] L. Csanky. Fast parallel matrix inversion algorithms. *SIAM J. Comput.*, 5(4):618–623, 1976.

[5] A. Silvescu D. Caragea and V. Honavar. A framework for learning from distributed data using sufficient statistics and its application to learning decision trees. *International Journal of Hybrid Intelligent Systems*, 2003.

[6] R. J. Williams D. E. Rumelhart, G. E. Hinton. Learning representation by back-propagating errors. In *Nature*, volume 323, pages 533–536, 1986.

[7] J. Dean and S. Ghemawat. Mapreduce: Simplified data processing on large clusters. *Operating Systems Design and Implementation*, pages 137–149, 2004.

[8] N.M. Dempster A.P., Laird and Rubin D.B.

[9] D.J. Frank. Power-constrained cmos scaling limits. *IBM Journal of Research and Development*, 46, 2002.

[10] P. Gelsinger. Microprocessors for the new millennium: Challenges, opportunities and new frontiers. In *ISSCC Tech. Digest*, pages 22–25, 2001.

[11] Leon Bottou Igor Durdanovic Hans Peter Graf, Eric Cosatto and Vladimire Vapnik. Parallel support vector machines: The cascade svm. In *NIPS*, 2004.

[12] J. Hartigan. *Clustering Algorithms*. Wiley, 1975.

[13] T. Hastie and R. Tibshirani. Discriminant analysis by gaussian mixtures. *Journal of the Royal Statistical Society B*, pages 155–176, 1996.

[14] R. Jin and G. Agrawal. Shared memory parallelization of data mining algorithms: Techniques, programming interface, and performance. In *Second SIAM International Conference on Data Mining,*, 2002.

[15] M. Kearns. Efficient noise-tolerant learning from statistical queries. pages 392–401, 1999.

[16] Michael Kearns and Umesh V. Vazirani. *An Introduction to Computational Learning Theory*. MIT Press, 1994.

[17] David Lewis. Naive (bayes) at forty: The independence asssumption in information retrieval. In *ECML98: Tenth European Conference On Machine Learning*, 1998.

[18] Kun Liu and Hillow Kargupta. Distributed data mining bibliography. *http://www.cs.umbc.edu/ hillol/DDMBIB/*, 2006.

[19] T. K. MOON. The expectation-maximization algorithm. In *IEEE Trans. Signal Process*, pages 47–59, 1996.

[20] G. Moore. Progress in digital integrated electronics. In *IEDM Tech. Digest*, pages 11–13, 1975.

[21] Wayne Iba Pat Langley and Kevin Thompson. An analysis of bayesian classifiers. In *AAAI*, 1992.

[22] John C. Platt. Fast training of support vector machines using sequential minimal optimization. pages 185–208, 1999.

[23] Daryl Pregibon. Logistic regression diagnostics. In *The Annals of Statistics*, volume 9, pages 705–724, 1981.

[24] T. Studt. There's a multicore in your future, http://tinyurl.com/ohd2m, 2006.

[25] Herb Sutter and James Larus. Software and the concurrency revolution. *Queue*, 3(7):54–62, 2005.

[26] L.G. Valiant. A theory of the learnable. *Communications of the ACM*, 3(11):1134–1142, 1984.

[27] V. Vapnik. *Estimation of Dependencies Based on Empirical Data*. Springer Verlag, 1982.

[28] R. E. Welsch and E. KUH. Linear regression diagnostics. In *Working Paper 173, Nat. Bur. Econ. Res.Inc*, 1977.

[29] K. Esbensen Wold, S. and P. Geladi. Principal component analysis. In *Chemometrics and Intelligent Laboratory Systems*, 1987.
